# A Non-parametric Multi-Scale Statistical Model for Natural Images

**Jeremy S. De Bonet & Paul Viola**
Artificial Intelligence Laboratory
Learning & Vision Group
545 Technology Square Massachusetts Institute of Technology
Cambridge, MA 02139

EMAIL: jsd@ai.mit.edu & viola@ai.mit.edu
HOMEPAGE: http://www.ai.mit.edu/projects/lv

## Abstract

The observed distribution of natural images is far from uniform. On the contrary, real images have complex and important structure that can be exploited for image processing, recognition and analysis. There have been many proposed approaches to the principled statistical modeling of images, but each has been limited in either the complexity of the models or the complexity of the images. We present a non-parametric multi-scale statistical model for images that can be used for recognition, image de-noising, and in a "generative mode" to synthesize high quality textures.

## 1 Introduction

In this paper we describe a multi-scale statistical model which can capture the structure of natural images across many scales. Once trained on example images, it can be used to recognize novel images, or to generate new images. Each of these tasks is reasonably efficient, requiring no more than a few seconds or minutes on a workstation.

The statistical modeling of images is an endeavor which reaches back to the 60's and 70's (Duda and Hart, 1973). Statistical approaches are alluring because they provide a unified view of learning, classification and generation. To date however, a generic, efficient and unified statistical model for natural images has yet to appear. Nevertheless, many approaches have shown significant competence in specific areas.

Perhaps the most influential statistical model for generic images is the Markov random field (MRF) (Geman and Geman, 1984). MRF's define a distribution over

images that is based on simple and local interactions between pixels. Though MRF's have been very successfully used for restoration of images, their generative properties are weak. This is due to the inability of the MRF's to capture long range (low frequency) interactions between pixels. Recently there has been a great deal of interest in hierarchical models such as the Helmholtz machine (Hinton et al., 1995; Dayan et al., 1995). Though the Helmholtz machine can be trained to discover long range structure, it is not easily applied to natural images.

Multi-scale wavelet models have emerged as an effective technique for modeling realistic natural images. These techniques hypothesize that the wavelet transform measures the underlying causes of natural images which are assumed to be statistically independent. The primary evidence for this conjecture is that the coefficients of wavelet transformed images are uncorrelated and low in entropy (hence the success of wavelet compression). These insights have been used for noise reduction (Donoho and Johnstone, 1993; Simoncelli and Adelson, 1996), and example driven texture synthesis (Heeger and Bergen, 1995). The main drawback of wavelet algorithms is the assumption of complete independence between coefficients. We conjecture that in fact there is strong cross-scale dependence between the wavelet coefficients of an image, which is consistent with observations in (De Bonet, 1997) and (Buccigrossi and Simoncelli, 1997).

## 2   Multi-scale Statistical Models

Multi-scale wavelet techniques assume that images are a linear transform of a collection of statistically independent random variables: $I = W^{-1}\vec{C}$, where $I$ is an image, $W^{-1}$ is the inverse wavelet transform, and $\vec{C} = \{c_k\}$ is a vector of random variable "causes" which are assumed to be independent. The distribution of each cause $c_k$ is $p_k(\cdot)$, and since the $c_k$'s are independent it follows that: $p(\vec{C}) = \prod_k p_k(c_k)$. Various wavelet transforms have been developed, but all share the same type of multi-scale structure — each row of the wavelet matrix $W$ is a spatially localized filter that is a shifted and scaled version of a single basis function.

The wavelet transform is most efficiently computed as an iterative convolution using a bank of filters. First a "pyramid" of low frequency downsampled images is created: $G_0 = I$, $G_1 = 2 \downarrow (g \otimes G_0)$, and $G_{i+1} = 2 \downarrow (g \otimes G_i)$, where $2 \downarrow$ downsamples an image by a factor of 2 in each dimension, $\otimes$ is the convolution operation, and $g$ is a low pass filter. At each level a series of filter functions are applied: $F_j^i = f_i \otimes G_j$, where the $f_i$'s are various types of filters. Computation of the $F_j^i$'s is a linear transformation that can thought of as a single matrix $W$. With careful selection of $g$ and $f_i$ this matrix can be constructed so that $W^{-1} = W^\top$ (Simoncelli et al., 1992)[1]. Where convenient we will combine the pixels of the feature images $F_j^i(x, y)$ into a single cause vector $\vec{C}$.

The expected distribution of causes, $c_k$, is a function of the image classes that are being modeled. For example it is possible to attempt to model the space of all natural images. In that case it appears as though the most accurate $p_k(\cdot)$'s are highly kurtotic which indicates that the $c_k$'s are most often zero but in rare cases take on very large values (Donoho and Johnstone, 1993; Simoncelli and Adelson, 1996). This is in direct contrast to the distribution of $c_k$'s for white-noise images – which is gaussian. The difference in these distributions can be used as the basis of noise reduction algorithms, by reducing the wavelet coefficients which are more

likely to be noise than signal.

Specific image classes can be modeled using similar methods (Heeger and Bergen, 1995)[2]. For a given set of input images the empirical distribution of the $c_k$'s is observed. To generate a novel example of a texture a new set of causes, $\vec{C}'$ is sampled from the assumed independent empirical distributions $p_k(\cdot)$. The generated images are computed using the inverse wavelet transform: $I' = W^{-1}\vec{C}'$. Bergen and Heeger have used this approach to build a probabilistic model of a texture from a single example image. To do this they assume that textures are *spatially ergodic* – that the expected distribution is not a function of position in the image. As a result the pixels in any one feature image, $F_j^i(x,y)$, are samples from the same distribution and can be combined[3].

Heeger and Bergen's work is at or near the current state of the art in texture generation. Figure 1 contains some example textures. Notice however, that this technique is much better at generating smooth or noise-like textures than those with well defined structure. Image structures, such as the sharp edges at the border of the tiles in the rightmost texture can not be modeled with their approach. These image features directly contradict the assumption that the wavelet coefficients, or causes, of the image are independent.

For many types of natural images the coefficients of the wavelet transform are not independent, for example images which contain long edges. While wavelets are local both in frequency and space, a long edge is not local in frequency nor in space. As a result the wavelet representation of such a feature requires many coefficients. The high frequencies of the edge are captured by many small high frequency wavelets. The long scale is captured by a number of larger low frequency wavelets. A model which assumes these coefficients are independent can never accurately model images which contain these non-local features. Conversely a model which captures the conditional dependencies between coefficients will be much more effective. We chose to approximate the joint distribution of coefficients as a chain, in which coefficients that occur higher in the wavelet pyramid condition the distribution of coefficients at lower levels (i.e. low frequencies condition the generation of higher frequencies).

For every pixel in an image define the *parent vector* of that pixel:

$$\vec{V}(x,y) = \left[ F_0^0(x,y), F_0^1(x,y), \ldots, F_0^N(x,y), \right.$$

$$F_1^0(\lfloor \tfrac{x}{2} \rfloor, \lfloor \tfrac{y}{2} \rfloor), F_1^1(\lfloor \tfrac{x}{2} \rfloor, \lfloor \tfrac{y}{2} \rfloor), \ldots, F_1^N(\lfloor \tfrac{x}{2} \rfloor, \lfloor \tfrac{y}{2} \rfloor), \ldots$$

$$\left. F_M^0(\lfloor \tfrac{x}{2^M} \rfloor, \lfloor \tfrac{y}{2^M} \rfloor), F_M^1(\lfloor \tfrac{x}{2^M} \rfloor, \lfloor \tfrac{y}{2^M} \rfloor), \ldots, F_M^N(\lfloor \tfrac{x}{2^M} \rfloor, \lfloor \tfrac{y}{2^M} \rfloor) \right] \quad (1)$$

where $M$ is the top level of the pyramid and $N$ is the number of features. Rather than generating each of these coefficients independently, we define a chain across scale. In this chain the generation of the lower levels depend on the higher levels:

$$p(\vec{V}(x,y)) = p(\vec{V}_M(x,y)) \times p(\vec{V}_{M-1}(x,y)|\vec{V}_M(x,y))$$

$$\times \; p(\vec{V}_{M-2}(x,y)|\vec{V}_{M-1}(x,y), \vec{V}_M(x,y)) \times \ldots$$

$$\times \; p(\vec{V}_0(x,y)|\vec{V}_1(x,y), \ldots, \vec{V}_{M-1}(x,y), \vec{V}_M(x,y)) \quad (2)$$

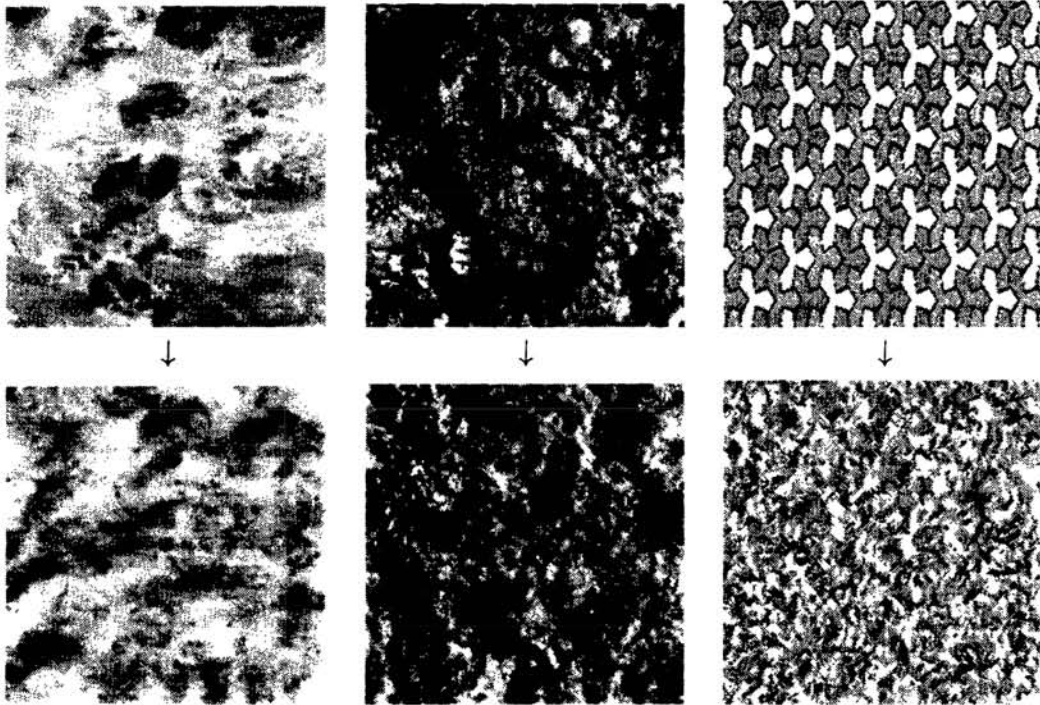

Figure 1: Synthesis results for the Heeger and Bergen (1995) model. TOP: Input textures. BOTTOM: Synthesis results. This technique is much better at generating fine or noisy textures then it is at generating textures which require co-occurrence of wavelets at multiple scales.

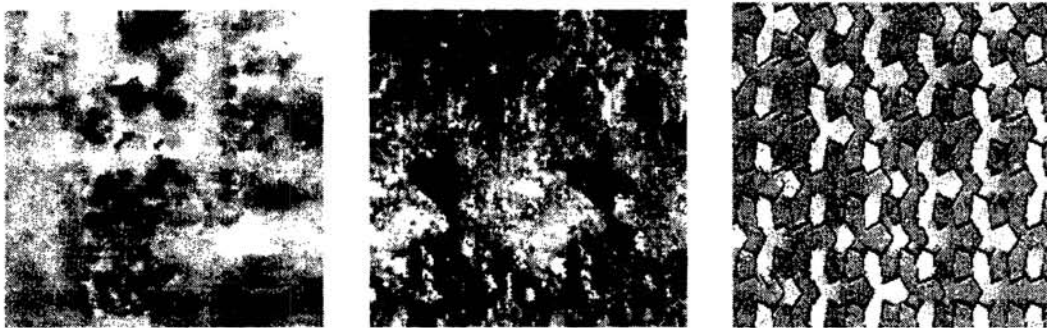

Figure 2: Synthesis results using our technique for the input textures shown in Figure 1 (TOP).

where $\vec{V}_l(x, y)$ is the a subset of the elements of $\vec{V}(x, y)$ computed from $G_l$. Usually we will assume ergodicity, i.e. that $p(\vec{V}(x, y))$ is independent of $x$ and $y$. The generative process starts from the top of the pyramid, choosing values for the $\vec{V}_M(x, y)$ at all points. Once these are generated the values at the next level, $\vec{V}_{M-1}(x, y)$, are generated. The process continues until all of the wavelet coefficients are generated. Finally the image is computed using the inverse wavelet transform.

It is important to note that this probabilistic model is not made up of a collection of independent chains, one for each $\vec{V}(x, y)$. Parent vectors for neighboring pixels have substantial overlap as coefficients in the higher pyramid levels (which are

lower resolution) are shared by neighboring pixels at lower pyramid levels. Thus, the generation of nearby pixels will be strongly dependent. In a related approach a similar arrangement of generative chains has been termed a Markov tree (Basseville et al., 1992).

## 2.1   Estimating the Conditional Distributions

The additional descriptive power of our generative model does not come without cost. The conditional distributions that appear in (2) must be estimated from observations. We choose to do this directly from the data in a non-parametric fashion. Given a sample of parent vectors $\left\{ \vec{S}(x,y) \right\}$ from an example image we estimate the conditional distribution as a ratio of Parzen window density estimators:

$$p(\vec{V}_l(x,y)|\vec{V}_{l+1}^M(x,y)) = \frac{p(\vec{V}_l^M(x,y))}{p(\vec{V}_{l+1}^M(x,y))} \approx \frac{\sum_{x',y'} R(\vec{V}_l^M(x,y), \vec{S}_l^M(x',y'))}{\sum_{x',y'} R(\vec{V}_{l+1}^M(x,y), \vec{S}_{l+1}^M(x',y'))} \quad (3)$$

where $\vec{V}_l^k(x,y)$ is a subset of the parent vector $\vec{V}(x,y)$ that contains information from level $l$ to level $k$, and $R(\cdot)$ is a function of two vectors that returns maximal values when the vectors are similar and smaller values when the vectors are dissimilar. We have explored various $R(\cdot)$ functions. In the results presented the $R(\cdot)$ function returns a fixed constant $1/z$ if all of the coefficients of the vectors are within some threshold $\theta$ and zero otherwise. Given this simple definition for $R(\cdot)$ sampling from $p(\vec{V}_l(x,y)|\vec{V}_{l+1}^M(x,y))$ is very straightforward: find all $x',y'$ such that $R(\vec{S}_{l+1}^M(x',y'), \vec{S}_{l+1}^M(x,y)) = 1/z$ and pick from among them to set $\vec{V}_l(x,y) = \vec{S}_l(x',y')$.

## 3   Experiments

We have applied this approach to the problems of texture generation, texture recognition, target recognition, and signal de-noising. In each case our results are competitive with the best published approaches.

In Figure 2 we show the results of our technique on the textures from Figure 1. For these textures we are better able to model features which are caused by a conjunction of wavelets. This is especially striking in the rightmost texture where the geometrical tiling is almost, but not quite, preserved. In our model, knowledge of the joint distribution provides constraints which are critical in the overall perceived appearance of the synthesized texture.

Using this same model, we can measure the textural similarity between a known and novel image. We do this by measuring the likelihood of generating the parent vectors in the novel image under the chain model of the known image. On "easy" data sets, such as the the MeasTex **Brodatz** texture test suite, performance is slightly higher than other techniques, our approach achieved 100% correct classification compared to 97% achieved by a gaussian MRF approach (Chellappa and Chatterjee, 1985). The MeasTex **lattice** test suite is slightly more difficult because each texture is actually a composition of textures containing different spatial frequencies. Our approach achieved 97% while the best alternate method, in this case Gabor Convolution Energy method (Fogel and Sagi, 1989) achieved 89%. Gaussian MRF's explicitly assume that the texture is a unimodal distribution and as a result achieve only 79% correct recognition. We also measured performance on a set of 20 types of natural texture and compared the classification power of this model to that of human observers (humans discriminate textures extremely accurately.) On this

Original                        Denoise Shrinkage               Shrinkage Residual

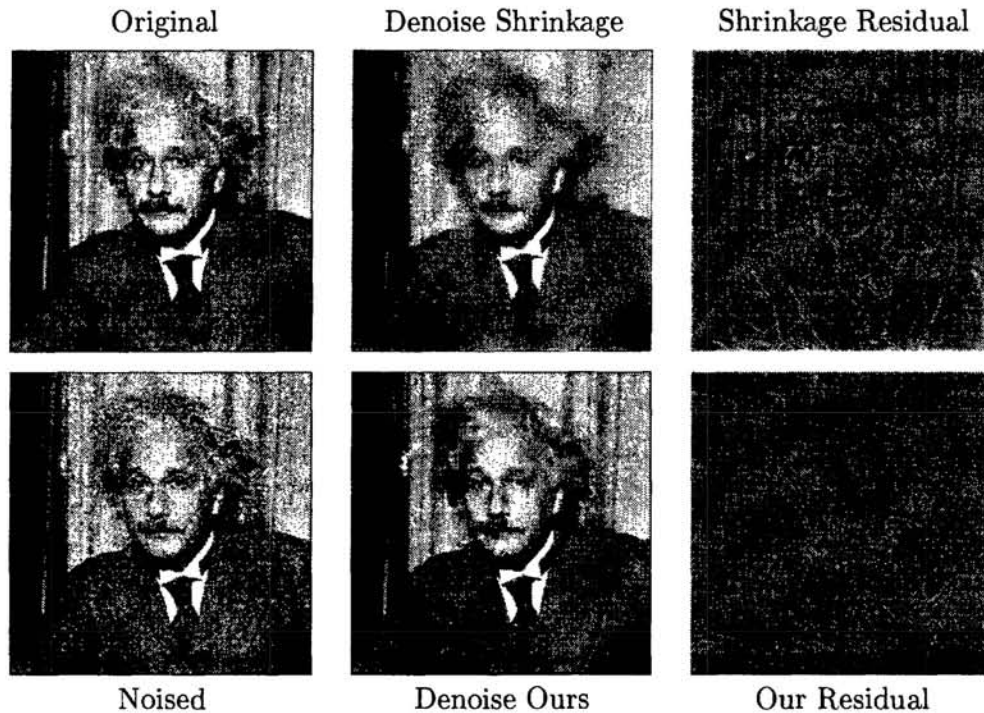

Noised                          Denoise Ours                    Our Residual

Figure 3: (Original) the original image; (Noised) the image corrupted with white gaussian noise (SNR 8.9 dB); (Denoise Shrinkage) the results of de-noising using wavelet shrinkage or coring (Donoho and Johnstone, 1993; Simoncelli and Adelson, 1996) (SNR 9.8 dB); (Shrinkage Residual) the residual error between the shrinkage de-noised result and the original — notice that the error contains a great deal of interpretable structure; (Denoise Ours) our de-noising approach (SNR 13.2 dB); and (Our Residual) the residual error — these errors are much less structured.

test, humans achieved 86% accuracy, our approach achieved an accuracy of 81%, and GMRF's achieved 68%.

A strong probabilistic model for images can be used to perform a variety of image processing tasks including de-noising and sharpening. De-noising of an observed image $\hat{I}$ can be performed by Monte Carlo averaging: draw a number of sample images according to the prior density $P(I)$, compute the likelihood of the noise for each image $P(\nu = \hat{(I)} - I)$, and then find the weighted average over these images. The weighted average is the estimated mean over all possible ways that the image might have been generated given the observation.

Image de-noising frequently relies on generic image models which simply enforce image smoothness. These priors either leave a lot of residual noise or remove much of the original image. In contrast, we construct a probability density model from the noisy image itself. In effect we assume that the image is redundant, containing many examples of the same visual structures, as if it were a texture. The value of this approach is directly related to the redundancy in the image. If the redundancy in the image is very low, then the parent structures will be everywhere different, and the only resampled images with significant likelihood will be the original image. But if there is some redundancy in the image — that might arise from a regular texture or smoothly varying patch — the resampling will freely average across these similar regions. This will have the effect of reducing noise in these images. In Figure 3 we show results of this de-noising approach.

# 4 Conclusions

We have presented a statistical model of texture which can be trained using example images. The form of the model is a conditional chain across scale on a pyramid of wavelet coefficients. The cross scale condtional distributions are estimated non-parametrically. This is important because many of the observed conditional distributions are complex and contain multiple modes. We believe that there are two main weaknesses of the current approach: i) the tree on which the distributions are defined are fixed and non-overlapping; and ii) the conditional distributions are estimated from a small number of samples. We hope to address these limitations in future work.

## Acknowledgments

In this research, Jeremy De Bonet is supported by the DOD Multidisciplinary Research Program of the University Research Initiative, and Paul Viola by Office of Naval Research Grant No. N00014-96-1-0311.

## Footnotes

[1]Computation of the inverse wavelet transform is algorithmically similar to the computation of the forward wavelet transform.

[2]See (Zhu, Wu and Mumford, 1996) for a related but more formal model.

[3]Their generation process is slightly more complex than this, involving a iteration designed to match the pixel histogram. The implementation used for generating the images in Figure 1 incorporates this, but we do not discuss it here.

# References

Basseville, M., Benveniste, A., Chou, K. C., Golden, S. A., Nikoukhah, R., and Willsky, A. S. (1992). Modeling and estimation of multiresolution stochastic processes. *IEEE Transactions on Information Theory*, 38(2):766–784.

Buccigrossi, R. W. and Simoncelli, E. P. (1997). Progressive wavelet image coding based on a conditional probability model. In *Proceedings ICASSP-97*, Munich, Germany.

Chellappa, R. and Chatterjee, S. (1985). Classification of textures using gaussian markov random fields. In *Proceedings of the International Joint Conference on Acoustics, Speech and Signal Processing*, volume 33, pages 959–963.

Dayan, P., Hinton, G., Neal, R., and Zemel, R. (1995). The helmholtz machine. *Neural Computation*, 7:1022–1037.

De Bonet, J. S. (1997). Multiresolution sampling procedure for analysis and synthesis of texture images. In *Computer Graphics*. ACM SIGGRAPH.

Donoho, D. L. and Johnstone, I. M. (1993). Adaptation to unknown smoothness via wavelet shrinkage. Technical report, Stanford University, Department of Statistics. Also Tech. Report 425.

Duda, R. and Hart, P. (1973). *Pattern Classification and Scene Analysis*. John Wiley and Sons.

Fogel, I. and Sagi, D. (1989). Gabor filters as texture discriminator. *Biological Cybernetics*, 61:103–113.

Geman, S. and Geman, D. (1984). Stochastic relaxation, gibbs distributions, and the bayesian restoration of images. *IEEE Transactions on Pattern Analysis and Machine Intelligence*, 6:721–741.

Heeger, D. J. and Bergen, J. R. (1995). Pyramid-based texture analysis/synthesis. In *Computer Graphics Proceedings*, pages 229–238.

Hinton, G., Dayan, P., Frey, B., and Neal, R. (1995). The "wake-sleep" algorithm for unsupervised neural networks. *Science*, 268:1158–1161.

Simoncelli, E. P. and Adelson, E. H. (1996). Noise removal via bayesian wavelet coring. In *IEEE Third Int'l Conf on Image Processing*, Laussanne Switzerland. IEEE.

Simoncelli, E. P., Freeman, W. T., Adelson, E. H., and Heeger, D. J. (1992). Shiftable multiscale transforms. *IEEE Transactions on Information Theory*, 38(2):587–607.

Zhu, S. C., Wu, Y., and Mumford, D. (1996). Filters random fields and maximum entropy(frame): To a unified theory for texture modeling. *To appear in Int'l Journal of Computer Vision*.